# Neural Implementation of Bayesian Inference in Population Codes

**Si Wu**
Computer Science Department
Sheffield University, UK

**Shun-ichi Amari**
Lab. for Mathematic Neuroscience,
RIKEN Brain Science Institute, JAPAN

## Abstract

This study investigates a population decoding paradigm, in which the estimation of stimulus in the previous step is used as prior knowledge for consecutive decoding. We analyze the decoding accuracy of such a Bayesian decoder (Maximum a Posteriori Estimate), and show that it can be implemented by a biologically plausible recurrent network, where the prior knowledge of stimulus is conveyed by the change in recurrent interactions as a result of Hebbian learning.

## 1 Introduction

Information in the brain is not processed by a single neuron, but rather by a population of them. Such a coding strategy is called population coding. It is conceivable that population coding has advantage of being robust to the fluctuation in a single neuron's activity. However, people argue that population coding may have other computationally desirable properties. One such property is to provide a framework for encoding complex objects by using basis functions [1]. This is inspired by the recent progresses in nonlinear function approximation, such as, sparse coding, overcomplete representation and kernel regression. These methods are efficient and show some interesting neuron-like behaviors [2,3]. It is reasonable to think that similar strategies are used in the brain under the support of population codes. However, to confirm this idea, a general suspicion has to be clarified: can the brain perform such complex statistic inference? An important work towards the answer of this question was done by Pouget and co-authors [4,5]. They show that Maximum Likelihood (ML) Inference, which is usually thought to be complex, can be implemented by a biologically plausible recurrent network using the idea of line attractor.

ML is a special case of Bayesian inference when the stimulus is (or assumed to be) uniformly distributed. In case there is prior knowledge on the stimulus distribution, Maximum a Posteriori (MAP) Estimate has better performance. Zhang et al. has successfully applied MAP for reconstructing the rat position in a maze from the activity of hippocampal place cells [6]. In their method, the prior knowledge is the rat's position in the previous time step, which restricts the variability of rat's position in the current step under the continuity constraint. It turns out that MAP has a much better performance than other decoding methods, and overcomes the inefficiency of ML when information is not sufficient (when the rat stops running).

This result implies that MAP may be used by the nervous system. So far, in the literature MAP has been mainly studied as a mathematic tool for reconstructing data, though its potential neural implementation was pointed out by [1,6].

In the present study, we will firmly show how to implement MAP in a biologic way. The same kind of recurrent network for achieving ML is used [4,5]. The decoding process consists of two steps. In the first step when there is no prior knowledge of the stimulus, the network implements ML. Its estimation is subsequently used to form the prior distribution of stimulus for consecutive decoding, which we assume is a Gaussian function with the mean value being the estimation. It turns out that this prior knowledge can be naturally conveyed by the change in the recurrent interactions according to the Hebbian learning rule. This is an interesting finding and suggests a new role of Hebbian learning. In the second step, with the changed interactions, the network implements MAP. The decoding accuracy of MAP and the optimal form of Gaussian prior are also analyzed in this paper.

## 2 MAP in Population Codes

Let us consider a standard population coding paradigm. There are $N$ neurons coding for a stimulus $x$. The population activity is denoted by $\mathbf{r} = \{r_i\}$. Here $r_i$ is the response of the $i$th neuron, which is given by

$$r_i = f_i(x) + \epsilon_i, \qquad (1)$$

where $f_i(x)$ is the tuning function and $\epsilon_i$ is a random noise.

The encoding process of a population code is specified by the conditional probability $Q(\mathbf{r}|x)$ (i.e., the noise model). The decoding is to infer the value of $x$ from the observed $\mathbf{r}$.

We consider a general Bayesian inference in a population code, which estimates the stimulus by maximizing a log posterior distribution, $\ln P(x|\mathbf{r})$, i.e.,

$$\begin{aligned} \hat{x} &= \mathrm{argmax}_x \quad \ln P(x|\mathbf{r}), \\ &= \mathrm{argmax}_x \quad \ln P(\mathbf{r}|x) + \ln P(x), \end{aligned} \qquad (2)$$

where $P(\mathbf{r}|x)$ is the likelihood function. It can be equal to or different from the real encoding model $Q(\mathbf{r}|x)$, depending on the available information of the encoding process [7]. $P(x)$ is the distribution of $x$, representing the prior knowledge. This method is also called Maximum a Posteriori (MAP). When the distribution of $x$ is, or is assumed to be (when there is no prior knowledge) uniform, MAP is equivalent to ML.

MAP could be used in the information processing of the brain in several occasions. Let us consider the following scenario: a stimulus is decoded in multiple steps. This happens when the same stimulus is presented through multiple steps, or during a single presentation, neural signals are sampled many times. In both cases, the brain successively gains a rough estimation of the stimulus in each step decoding, which can serve to be the prior knowledge when further decoding is concerned. It is therefore natural to use MAP in this situation. Experiencing slightly different stimuli in consecutive steps as studied in [6], or more generally, stimulus slowly changes with time (multiple-step diagram is a discreted approximation), is a similar scenario. For simplicity, we only consider that stimulus is unchanged in the present study.

## 2.1 The Performance of MAP

Let us analyze the performance of MAP. Some notations are introduced first. Denote $\hat{x}_t$ a particular estimation of the stimulus in the $t$th step, and $\Omega_t^2$ the corresponding variance. The prior distribution of $x$ in the $t + 1$th step is assumed to be a Gaussian with the mean value $\hat{x}_t$, i.e.,

$$P(x|\hat{x}_t) = \frac{1}{\sqrt{2\pi}\tau_t} \exp^{-(x-\hat{x}_t)^2/2\tau_t^2}, \tag{3}$$

where the parameter $\tau_t$ reflects the estimator's confidence on $\hat{x}_t$, whose optimal value will be calculated later.

The posterior distribution of $x$ in the $t + 1$th step is given by

$$P(x|\mathbf{r}) = \frac{P(\mathbf{r}|x)P(x|\hat{x}_t)}{P(\mathbf{r})}, \tag{4}$$

and the solution of MAP is obtained by solving

$$\begin{aligned} \nabla \ln P(\hat{x}_{t+1}|\mathbf{r}) &= \nabla \ln P(\mathbf{r}|\hat{x}_{t+1}) - (\hat{x}_{t+1} - \hat{x}_t)/\tau_t^2, \\ &= 0. \end{aligned} \tag{5}$$

We calculate the decoding accuracies iteratively. In the first step decoding, since there is no prior knowledge on $x$, ML is used, whose decoding accuracy is known to be [7]

$$\Omega_1^2 = \frac{< (\nabla \ln P(\mathbf{r}|x))^2 >}{< -\nabla\nabla \ln P(\mathbf{r}|x) >^2}, \tag{6}$$

where the bracket $< \cdot >$ denotes averaging over $Q(\mathbf{r}|x)$.

Note that, to get the above result, we have considered that ML is asymptotically or quasi-asymptotically (when an unfaithful model is used) efficient [7]. This includes the cases when neural responses are independent, weakly correlated, uniformly correlated, correlated with strength proportional to firing rate (multiplicative correlation), or the fluctuation in neural responses are sufficiently small. In other strong correlation cases, ML is proved to be non-Fisherian, i.e, its decoding error satisfies a Cauchy type of distribution with variance diverging. Decoding accuracy can no longer be quantified by variance in such situations (for details, please refer to [8]).

Now come to calculate the decoding error in the second step. Suppose $\hat{x}_2$ is close enough to $x$. By expanding $\nabla \ln P(\mathbf{r}|\hat{x}_2)$ at $x$ in eq.(5), we obtain

$$\nabla \ln P(\mathbf{r}|x) + \nabla\nabla \ln P(\mathbf{r}|x)(\hat{x}_2 - x) - (\hat{x}_2 - \hat{x}_1)/\tau_1^2 = 0. \tag{7}$$

The random variable $\hat{x}_1$ can be decomposed as $\hat{x}_1 = x + \epsilon_1$, where $\epsilon_1$ is a random number satisfying Gaussian distribution of zero mean and variance $\Omega_1^2$.

By using the notation of $\epsilon_1$, we have

$$\hat{x}_2 - x = \frac{\nabla \ln P(\mathbf{r}|x) + \epsilon_1/\tau_1^2}{1/\tau_1^2 - \nabla\nabla \ln P(\mathbf{r}|x)}. \tag{8}$$

For the correlation cases considered in the present study (i.e, those ensure ML asymptotically or quasi-asymptotically efficient), $-\nabla\nabla \ln P(\mathbf{r}|x)$ can be approximated as a (positive) constant according to the law of large numbers [7,8]. Therefore, we can define a constant variable

$$\alpha = \tau_1^2(-\nabla\nabla \ln P(\mathbf{r}|x)), \tag{9}$$

and a random variable

$$R = \frac{\nabla \ln P(\mathbf{r}|x)}{-\nabla\nabla \ln P(\mathbf{r}|x)}. \tag{10}$$

Obviously $R$ satisfies the Gaussian distribution of zero mean and variance $\Omega_1^2$.

By using the notations $\alpha$ and $R$, we get

$$\hat{x}_2 - x = \frac{\alpha R + \epsilon_1}{1 + \alpha} \tag{11}$$

whose variance is calculated to be

$$\Omega_2^2 = \frac{1 + \alpha^2}{(1 + \alpha)^2}\Omega_1^2. \tag{12}$$

Since $(1 + \alpha^2)/(1 + \alpha)^2 \leq 1$ holds for any positive $\alpha$, the decoding accuracy in the second step is always improved. It is not difficult to check that its minimum value is

$$\Omega_2^2 = \frac{1}{2}\Omega_1^2, \tag{13}$$

when $\alpha = 1$, or, the optimal value of $\tau_1^2$ is

$$\tau_1^2 = \frac{1}{-\nabla\nabla \ln P(\mathbf{r}|x)} \tag{14}$$

When a faithful model is used, $-\nabla\nabla \ln Q(\mathbf{r}|x)$ is the Fisher information. $\tau_1^2$ hence equals to the variance of decoding error. This is understandable.

Following the same procedure, it can be proved that the optimal decoding accuracy in the $t$th step is $\Omega_t^2 = \frac{1}{t}\Omega_1^2$ when the width of Gaussian prior being $\tau_t^2 = \frac{1}{t}\tau_1^2$.

It is interesting to see that the above multiple decoding procedure, when the optimal values of $\tau_t$ are used, achieves the same decoding accuracy as a one-step ML by using all $N \times t$ signals. This is the best for any estimator to achieve. However, the multiple decoding is not a trivial replacement of one-step ML, and has many advantages. One of them is to save memory, considering that only $N$ signals and the value of previous estimation are stored in each step. Moreover, if a slowly changing stimulus is concerned, the multiple decoding outperforms one-step ML for the balance between adaptation and memory. These properties are valuable when information is processed in the brain.

## 3  Network Implementation of MAP

In this section, we investigate how to implement MAP by a recurrent network. A two-step decoding is studied. Without loss of generality, we consider $N \longrightarrow \infty$ and do calculation in the continuous limit.

The network we consider is a fully connected one-dimensional homogeneous neural field, in which $c$ denotes the position coordinate, i.e., the neurons' preferred stimuli. The tuning function of the neuron with preferred stimulus $c$ is

$$f_c(x) = \frac{1}{\sqrt{2\pi}a} \exp^{-(c-x)^2/2a^2}. \tag{15}$$

For simplicity, we consider an encoding process in which the fluctuations in neurons' responses are independent Gaussian noises (more general correlated cases can be handled similarly), that is,

$$Q(\mathbf{r}|x) = \frac{1}{Z} \exp^{-\frac{\rho}{2\sigma^2} \int (r_c - f_c(x))^2 dc}, \tag{16}$$

where $\rho$ is the neuron density and $Z$ is the normalization factor. A faithful model is used in both steps decoding, i.e., $P(\mathbf{r}|x) = Q(\mathbf{r}|x)$ (again, generalization to more general cases of $P(\mathbf{r}|x) \neq Q(\mathbf{r}|x)$ is straightforward.).

For the above model setting, the solution of ML in the first step is calculated to be

$$\hat{x}_1 = \text{argmax}_x \quad \int r_c f_c(x) dc, \qquad (17)$$

where the condition $\int f_c^2(x) dc = \text{const}$ has been used.

The solution of MAP in the second step is

$$\hat{x}_2 = \text{argmax}_x \quad \int r_c f_c(x) dc - (x - \hat{x}_1)^2 / 2\tau_1^2. \qquad (18)$$

Compared with eq.(17), eq.(18) has one more term corresponding to the contribution of prior distribution.

Now come to the study of using a recurrent network to realize eqs.(17) and (18). Following the idea of Pouget et al. [4,5], the following network dynamics is constructed. Let $U_c$ denote the (average) internal state of neuron at $c$, and $W_{c,c'}$ the recurrent connection weights from neurons at $c$ to those at $c'$. The dynamics of neural excitation is governed by

$$\frac{dU_c}{dt} = -U_c + \int W_{c,c'} O_{c'} dc' + I_c, \qquad (19)$$

where

$$O_c = \frac{U_c^2}{1 + \mu \int U_c^2 dc} \qquad (20)$$

is the activity of neurons at $c$ and $I_c$ is the external input arriving at $c$.

The recurrent interactions are chosen to be

$$W_{c,c'} = \exp^{-(c-c')^2/2a^2}, \qquad (21)$$

which ensures that when there is no external input ($I_c = 0$), the network is neutrally stable on line attractor,

$$\tilde{O}_c(z) = D \exp^{-(c-z)^2/2a^2} \qquad \forall z, \qquad (22)$$

where the parameter $D$ is constant and can be determined easily. Note that the line attractor has the same shape as the tuning function. This is crucial, which allows the network perform template-matching by using the tuning function, being as same as ML and MAP.

When a sufficiently small input $I_c$ is added, the network is no longer neutrally stable on the line attractor. It can be proved that the steady state of the network has approximately the same shape as eq.(22) (the deviation is of the 2nd order of the magnitude of $I_c$.), whereas, its steady position on the line attractor (i.e., the network estimation) is determined by maximizing the overlap between $I_c$ and $\tilde{O}_c(z)$ [4,9].

Thus, if $I_c = \varepsilon r_c$ in the first step[1], where $\varepsilon$ is a sufficiently small number, the network estimation is given by

$$\hat{z}_1 = \text{argmax}_z \quad \int r_c \tilde{O}_c(z) dc, \qquad (23)$$

which has the same value as the solution of ML (see eq.(17)). We say that the network implements ML.

To implement MAP in the second step, it is critical to identify a neural mechanism which can 'transmit' the prior knowledge obtained in the first step to the second one. We find that this is naturally done by Hebbian learning.

After the first step decoding, the recurrent interaction changes a small amount according to the Hebbian rule, whose new value is

$$\tilde{W}(c,c') = W_{c,c'} + \eta \tilde{O}_c(\hat{z}_1)\tilde{O}_{c'}(\hat{z}_1), \tag{24}$$

where $\eta$ is a small positive number representing the Hebbian learning rate, and $\tilde{O}_c(\hat{z}_1)$ is the neuron activity in the first step.

With the new recurrent interactions, the net input from other neurons to the one at $c$ is calculated to be

$$
\begin{aligned}
\int \tilde{W}_{c,c'} O_{c'} dc' &= \int W_{c,c'} O_{c'} dc' + \eta \tilde{O}_c(\hat{z}_1) \int O_{c'}(\hat{z}_1) O_{c'} dc', \\
&\approx \int W_{c,c'} O_{c'} dc' + \nu \tilde{O}_c(\hat{z}_1),
\end{aligned}
\tag{25}
$$

where $\nu$ is a small constant. To get the last approximation, the following facts have been used: 1) The initial state of neuron in the second step is at $\tilde{O}_c(\hat{z}_1)$, 2) The neuron activity $O_c$ during the second step is between $\tilde{O}_c(\hat{z}_1)$ and $\tilde{O}_c(\hat{z}_2)$, where $\hat{z}_2$ is the position of the steady state; 3) $(\hat{z}_1 - \hat{z}_2)^2/2a^2 \ll 1$, considering that neurons are widely tuned as seen in data ($a$ is large) and consecutive estimations are close enough. These factors ensures the approximation, $\int \tilde{O}_{c'}(\hat{z}_1) O_{c'} dc' \approx$ const to be good enough.

Substituting eq.(25) in (19), we see that the network dynamics in the second step, when compared with the first one, is in effect to modify the input $I_c$ to be $I_c' = \varepsilon(r_c + A\tilde{O}_c(\hat{z}_1))$, where $A$ is a constant and can be determined easily.

Thus, the network estimation in the second step is determined by maximizing the overlap between $I_c'$ and $\tilde{O}_c(z)$, which gives

$$\hat{z}_2 = \text{argmax}_z \int r_c \tilde{O}_c(z) dc + A \int \tilde{O}_c(\hat{z}_1)\tilde{O}_c(z) dc. \tag{26}$$

The first term in the right handside is known to achieve ML. Let us see the contribution of the second one, which can be transformed to

$$
\begin{aligned}
\int \tilde{O}_c(\hat{z}_1)\tilde{O}_c(z) dc &= B \exp^{-(\hat{z}_1-z)^2/4a^2}, \\
&\approx -B(z-\hat{z}_1)^2/4a^2 + \text{terms not on z,}
\end{aligned}
\tag{27}
$$

where $B$ is a constant. Again, in the above calculation, $(\hat{z}_1 - z)^2/4a^2 \ll 1$ is used for the same argument discussed above.

Compare eqs.(18) and (27), we see that the second term plays the same role as the prior knowledge in MAP. Thus, the network indeed implements MAP. The value of $A$ (or the Hebbian learning rate) can be adjusted accordingly to match the optimal choice of $\tau_1^2$.

The above result is confirmed by the simulation experiment (Table.1), which was done with 101 neurons uniformly distributed in the region $[-3,3]$ and the true stimulus being at 0. It shows that the estimation of the network agrees well with MAP.

Table 1: Comparing the decoding accuracies of the network and MAP with different values of $\alpha$ (the corresponding values of $\tau_1^2$ and $A$ are adjusted.). The parameters are $a = 1$, $\mu = 0.5$ and $\sigma^2 = 0.01$. The data is obtained after 100 trials.

| $\alpha$ | 0.1 | 0.5 | 1 | 2 | 5 |
|---|---|---|---|---|---|
| MAP | 0.000250 | 0.000167 | 0.000150 | 0.000167 | 0.000217 |
| Network | 0.000263 | 0.000178 | 0.000161 | 0.000184 | 0.000227 |

## 4 Conclusion and Discussion

In summary we have investigated how to implement MAP by using a biologically plausible recurrent network. A two-step decoding paradigm is studied. In the first step when there is no prior knowledge, the network implements ML, whose estimation is subsequently used to form the prior distribution of stimulus for consecutive decoding. In the second step, the network implements MAP.

Line attractor and Hebbian learning are two critical elements to implement MAP. The former enables the network to do template-matching by using the tuning function, being as same as ML and MAP. The latter provides a mechanism that conveys the prior knowledge obtained from the first step to the second one. Though the results in this paper may quantitatively depend on the formulation of the models, it is reasonable to believe that they are qualitatively true, as both Hebbian learning and line attractor are biologically plausible. Line attractor comes from the translation invariance of network interactions, and has been shown to be involved in several neural computations [10-12]. We expect that the essential idea of Bayesian inference of utilizing previous knowledge for successive decoding is used in the information processing of the brain.

We also analyzed the decoding accuracy of MAP in a population code and the optimal form of Gaussian prior. In the present study, stimulus is kept to be fixed during consecutive decodings. A generalization to the case when stimulus slowly changes over time is straightforward.

**References**

[1] A. Pouget, P. Dayan & R. Zemel. *Nature Reviews Neuroscience*, **1**, 125-132, 2000.

[2] B. Olshausen & D. Field. *Nature*, **381**, 607-609, 1996.

[3] T. Poggio & F. Girosi. *Neural Computation*, **10**, 1445-1454, 1998.

[4] A. Pouget & K. Zhang. *NIPS*, **9**, 1997.

[5] S. Deneve, P. E. Latham & A. Pouget. *Nature Neuroscience*, **2**, 740-745, 1999.

[6] K. Zhang, I. Ginzburg, B. McNaughton & T. Sejnowski. *J. Neurophysiol.*, **79**, 1017-1044, 1998.

[7] S. Wu, H. Nakahara & S. Amari. *Neural Computation*, **13**, 775-798, 2001.

[8] S. Wu, S. Amari & H. Nakahara. *CNS*01* (to appear).

[9] S. Wu, S. Amari & H. Nakahara. *Neural Computation* (in press).

[10] S. Amari. *Biological Cybernetics*, **27**, 77-87, 1977.

[11] K. Zhang. *J. Neurosci.*, **16**, 2112-2126, 1996.

[12] H. Seung. *Proc. Natl. Acad. Sci. USA*, **93**, 13339-13344, 1996.

## Footnotes

[1] Consider an instant input, triggering the network to be initially at $O_c(t = 0) = r_c$, as used in [5], has the same result.
